# Avoiding False Positive in Multi-Instance Learning

**Yanjun Han, Qing Tao, Jue Wang**
Institute of Automation, Chinese Academy of Sciences
Beijing, 100190, China
`yanjun.han, qing.tao, jue.wang@ia.ac.cn`

## Abstract

In multi-instance learning, there are two kinds of prediction failure, i.e., false negative and false positive. Current research mainly focus on avoiding the former. We attempt to utilize the geometric distribution of instances inside positive bags to avoid both the former and the latter. Based on kernel principal component analysis, we define a projection constraint for each positive bag to classify its constituent instances far away from the separating hyperplane while place positive instances and negative instances at opposite sides. We apply the Constrained Concave-Convex Procedure to solve the resulted problem. Empirical results demonstrate that our approach offers improved generalization performance.

## 1 Introduction

Multi-instance Learning (MIL) was first proposed by Dietterich et.al. in [1] to predict the binding ability of a drug from its biochemical structure. A certain drug molecule corresponds to a set of conformations which cannot be differentiated via chemical experiments. A drug is labeled positive if any of its constituent conformations has the binding ability greater than the threshold, otherwise negative. Therefore, each sample (a drug) is a bag of instances (its constituent conformations). In multi-instance learning the label information for positive samples is incomplete in that the instances in a certain positive bag are all labeled positive. Generally, methods for multi-instance learning are modified versions of approaches for supervised learning by shifting the focus from discrimination on instances to discrimination on bags.

The earliest exploration were the APR algorithms proposed in [1]. From then on, a number of approaches emerged. Examples include Diverse Density [2], Citation $k-$NN [3], MI-SVMs [4], MI-kernels [5], reg-SVM [6], MissSVM [7], sbMIL, stMIL [8], PPMM [9], MIGraphs [10], etc. Many real-world applications can be regarded as Multi-instance learning problems. Examples include image classification [11], document categorization [12], computer aided diagnosis [13], etc.

As far as positive bags are concerned, current research usually treat them as labyrinths in which witnesses (responsible positive instances) are encaged, and consider nonwitnesses (other instances) therein to be useless or even distractive. The information carried by nonwitnesses is not well utilized. Factually, nonwitnesses are indispensable for characterizing the overall instance distribution, and thus help to improve the learner. Several researchers realized the importance of nonwitnesses and attempted to utilize them. In MI-kernels [5] and reg-SVM [6], nonwitnesses together with witnesses are squeezed into the kernel matrix. In mi-SVM [4], the labels of all nonwitnesses are treated as unknown integer variables to be optimized. mi-SVM tends to misclassify negative instances in positive bags since the resulted margin will be larger. And we will elaborate on this flaw in section 3.1. In MissSVM [7] and stMIL [8], multi-instance learning is addressed from the view of semi-supervised learning, and nonwitnesses are treated as unlabeled data, whose labels should be assigned to maximize the margin. sbMIL [8] attempt to estimate the ratio of positive instances inside positive bags and utilize this information in the subsequent classification. MissSVM, sbMIL and stMIL suffer from the same flaw as mi-SVM.

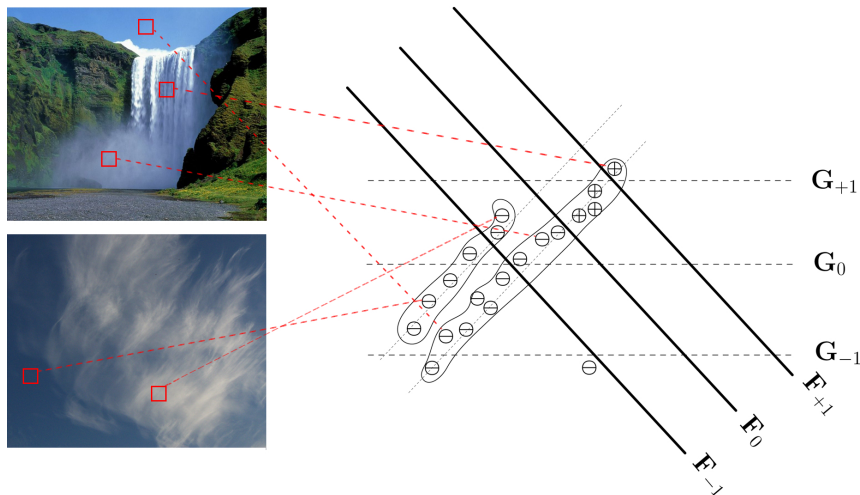

Figure 1: Illustration of the False Positive Phenomenon: The top image is a positive training sample, and the bottom image is a negative testing sample. The symbol $\oplus$ and $\ominus$ respectively denote positive and negative instances. Enveloped points are instances in a positive bag. The Point not enveloped is a negative bag of just one instance. Separating plane $\mathbf{F}_i$ corresponds to $f(\mathbf{x}) = i$, and $\mathbf{G}_i$ corresponds to $g(\mathbf{x}) = i$. The learners $f$ and $g$ are obtained with and without the projection constraint, respectively. Instances are labeled according to $f$. For details, please refer to the passage below.

The neglect of nonwitnesses in positive bags may lead to false positive and cause a model to misclassify unseen negative samples. For example, in natural scene classification, each image is segmented to a bag of instances beforehand, and each instance is a patch (ROI, Regions Of Interest) characterized by one feature vector describing its color. The task is to predict whether an image contains a waterfall or not (Figure 1). A positive image contains some positive instances corresponding to waterfall and some negative instances from other categories such as sky, stone, grass, etc., while a negative bag exclusively contains negative instances from other categories. Naturally, some negative instances (patches) only exist in positive bags. For instance, the end of a waterfall is often surrounded by mist. The aforementioned approaches tend to misclassify negative instances in positive bags. Therefore, the patch corresponding to mist is misclassified as positive. Given an unseen image with cirrus cloud and without waterfall, the obtained learner will misclassify this image as positive because cirrus cloud and mist are similar to each other.

To avoid both false negative and false positive, we attempt to classify instances inside positive bags far from the separating hyperplane and place positive and negative instances at opposite sides. We achieve this by introducing projection constraints based on kernel principal component analysis into MI-SVM [4]. Each constraint is defined on a positive bag to encourage large variance of its constituent instances along the normal direction of the separating hyperplane. We apply the Constrained Concave-Convex Procedure (CCCP) to solve the resulted optimization problems.

The remainder of the paper is organized as follows: Section 2 introduces notation convention and the CCCP. In Section 3 we bring out the projection constraint and the corresponding formulation for multi-instance learning. In Section 4, the algorithm is evaluated on real world data sets. Finally, conclusions are drawn in Section 5.

## 2 Preliminaries

### 2.1 Notation Convention

The origin of multi-instance learning [1] has been presented in section 1. Let $\mathcal{X} \subseteq \mathbb{R}^p$ be the space containing instances and $D = \{(B_i, y_i)\}_{i=1}^m$ be the training data, where $B_i$ is the $i^{th}$ bag of instances $\{\mathbf{x}_{i1}, \cdots, \mathbf{x}_{in_i}\}$ and $y_i \in \mathcal{Y}$ is the label for $B_i$. $\mathcal{Y}$ is $\{+1, -1\}$ for classification and $\mathbb{R}$ for regression. In addition, denote the index set for instances $\mathbf{x}_{ij}$ of $B_i$ by $\mathcal{I}_i$. The task is to train

a learner to predict the label of an unseen bag. Compared with traditional supervised learning, the learner is a mapping from $2^{\mathcal{X}}$ to $\mathcal{Y}$ instead of from $\mathcal{X}$ to $\mathcal{Y}$. Denote the index sets for positive and negative bags by $\mathcal{I}_+$ and $\mathcal{I}_-$ respectively. Without loss of generality, assume that the instances are ordered in the sequence $\{\mathbf{x}_{11}, \cdots, \mathbf{x}_{1n_1}, \cdots, \mathbf{x}_{m1}, \cdots, \mathbf{x}_{mn_m}\}$. We index instances by a function $I(\mathbf{x_{ij}}) = \sum_{k=1}^{i-1} n_k + j$. And $I(B_i)$ returns a vector $(\sum_{k=1}^{i-1} n_k + 1, \cdots, \sum_{k=1}^{i-1} n_k + n_i)$.

## 2.2 Constrained Concave-Convex Procedure

Non-convex optimizations are undesirable because few algorithms effectively converge even to a local optimum. However, if both objective function and constraints take the form of a difference between two convex functions, then a non-convex problem can be solved efficiently by the constrained concave-convex procedure [14]. The fundamental is to eliminate the non-convexity by changing non-convex parts to their first-order Taylor expansions. The original problem is as follows:

$$\min_{\mathbf{x}} f_0(\mathbf{x}) - g_0(\mathbf{x})$$
$$s.t. \ f_i(\mathbf{x}) - g_i(\mathbf{x}) \leq c_i, \quad i = 1, \cdots, m \tag{1}$$

where $f_i, g_i (i = 0, \cdots, m)$ are real-valued, convex and differentiable functions on $R^n$. Starting from a random $\mathbf{x}^{(0)}$, (1) is approximated by a sequence of successive convex optimization problems. At the $t + 1^{th}$ iteration, the non-convex parts in the objective and constraints are substituted by their first-order Taylor expansions, and the resulted optimization problem is as follows:

$$\min_{\mathbf{x}} f_0(\mathbf{x}) - \left[ g_0(\mathbf{x^{(t)}}) + \nabla g_0(\mathbf{x^{(t)}})^T (\mathbf{x} - \mathbf{x^{(t)}}) \right] \tag{2}$$
$$s.t. \ f_i(\mathbf{x}) - \left[ g_i(\mathbf{x^{(t)}}) + \nabla g_i(\mathbf{x^{(t)}})^T (\mathbf{x} - \mathbf{x^{(t)}}) \right] \leq c_i$$

where $\mathbf{x^{(t)}}$ is the optimal solution to (2) at the $t^{th}$ iteration. The above procedure is repeated until convergence. In [14] it is proved that the CCCP converges to a local optimum of (1).

## 3 Multi-Instance Classification

### 3.1 Support Vector Machine Formulation

Our work is based on the support vector machine (SVM) formulations for multi-instance learning, to be exact, the MI-SVM [4] as follows:

$$\min_{\mathbf{w},b,\boldsymbol{\xi}} \frac{1}{2} \|\mathbf{w}\|^2 + C \Big[ \sum_{i \in \mathcal{I}_+} \xi_i + \sum_{j \in \mathcal{I}_i, i \in \mathcal{I}_-} \xi_{ij} \Big] \tag{3}$$
$$s.t. \ \max_{j \in \mathcal{I}_i} (\mathbf{w}^T \mathbf{x}_{ij} + b) \geq 1 - \xi_i, \ \xi_i \geq 0, \ i \in \mathcal{I}_+$$
$$- \mathbf{w}^T \mathbf{x}_{ij} - b \geq 1 - \xi_{ij}, \ \xi_{ij} \geq 0, \ j \in \mathcal{I}_i, i \in \mathcal{I}_-$$

Compared with the conventional SVM, in MI-SVM the notion of slack variables for positive samples is extended from individual instances to bags while that for negative samples remains unchanged. As shown by the first set of max constraints, only the "most positive" instance in a positive bag, or the witness, could affect the margin. And other instances, or nonwitnesses, become irrelevant for determining the position of the separating plane once the witness is specified.

The max constraint at first sight seems to well embody the characteristic of multi-instance learning. Indeed, it helps to avoid the false negative, i,e., the misclassification of positive samples. However, it may incur false positive due to the following two reasons. Firstly, the max constraint aims at discovering the witness, and tends to skip nonwitnesses. Thus each positive bag is approximately oversimplified to a single pattern, i.e., the witness. Most information in positive bags is wasted. Secondly, due to the characteristic of the max function and the greediness of optimization methods, the predictions of nonwitnesses are often adjusted above zero in the learning process. Besides, there is no mechanism to draw the predictions of nonwitenesses below zero. Nevertheless, many nonwitnesses in positive bags are factually negative instances. For example, in natural scene classification,

many image patches in a positive bag are from the irrelevant background; in document categorization, many posts in a positive bag are not from the target category. Hence, many nonwitnesses are mislabeled as positive, and we obtain a falsely large margin.

As shown in Figure 1, MI-SVM classifies half instances in the training sample as positive, and some negative instances are mislabeled. This false positive will impair the generalization performance.

## 3.2 Projection Constraint

The above problem is not unique for MI-SVM. Any approach without properly utilizing nonwitnesses has the same problem. In our preliminary work before this paper, we tried three solutions. Firstly, we treat the labels of all nonwitnesses as unknown integer variables to be optimized. In the SVM framework, it is exactly the mi-SVM [4] as follows:

$$\min_{\{y_{ij}\}} \min_{\mathbf{w},b,\boldsymbol{\xi}} \frac{1}{2}\|\mathbf{w}\|^2 + C\sum_{j\in\mathcal{I}_i,\ i\in\mathcal{I}_+\cup\mathcal{I}_-}\xi_{ij} \tag{4}$$
$$s.t.\ y_{ij}(\mathbf{w}^T\mathbf{x}_{ij}+b) \geq 1-\xi_{ij},\ \xi_{ij}\geq 0,\ j\in\mathcal{I}_i,\ i\in\mathcal{I}_+$$
$$\sum_{j\in\mathcal{I}_i}\frac{y_{ij}+1}{2} \geq 1, \qquad\qquad i\in\mathcal{I}_+$$
$$-\mathbf{w}^T\mathbf{x}_{ij}-b \geq 1-\xi_{ij},\ \xi_{ij}\geq 0,\ j\in\mathcal{I}_i,\ i\in\mathcal{I}_-$$

It seems that assigning labels over all nonwitnesses should lead to a reasonable model. Nevertheless, nonwitnesses are usually labeled positive since the consequent margin will be larger. Thus, many of nonwitnesses are misclassified. As far as the example in Figure 1 is concerned, the obtained learner is $g(\mathbf{x})$ instead of $f(\mathbf{x})$. MissSVM [7] takes an unsupervised approach. For every instance in positive bags, two slack variables are introduced, measuring the distances from the instance to the positive boundary $f(\mathbf{x}) = +1$ and the negative boundary $f(\mathbf{x}) = -1$ respectively, and the label of the instance depends on the smaller slack variable. stMIL [8] takes a similar approach. As mi-SVM, MissSVM and stMIL also suffers from misclassification of nonwitnesses. sbMIL [8] tackles multi-instance learning in two steps. The first step is similar to MI-SVM, and the second step is a traditional SVM. Still, there is no mechanism in sbMIL to avoid false positive.

In the second solution, we simultaneously seek for the "most positive" instance and the "most negative" instance in a positive bag by adding the following constraints to (3):

$$(-1)\cdot\min_{j\in\mathcal{I}_i}(\mathbf{w}^T\mathbf{x}_{ij}+b) \geq -1-\zeta_i,\ \zeta_i\geq 0,\ i\in\mathcal{I}_+ \tag{5}$$

And the term $\sum_{i\in\mathcal{I}_+}\xi_i$ in the objective of (3) is changed to $\sum_{i\in\mathcal{I}_+}(\xi_i+\zeta_i)$. Although misclassification of nonwitnesses is alleviated since at least the "most negative" nonwitness is classified correctly, the information carried by most nonwitnesses are not fully utilized. As far as the example in Figure 1 is concerned, the obtained learner is still $g(\mathbf{x})$ instead of $f(\mathbf{x})$. Besides, this solution is not appropriate for applications which involve positive bags only with positive instances.

The third solution is the projection constraint proposed in this paper. In a maximum margin framework we want to classify instances in a positive bag far away from the separating hyperplane while place positive instances and negative instances at opposite sides. From another point of view, in the feature (kernel) space, we want to maximize the variance of instances in a positive bag along $\mathbf{w}$, the normal vector of the separating hyperplane. Therefore, the principal component analysis (PCA) [15] is just the technique that we need. To tackle complicated real world datasets, we directly develop our approach in the Reproducing Kernel Hilbert Space (RKHS). Let $\mathcal{X}$ be the space of instances, and $\mathcal{H}$ be a RKHS of functions $f : \mathcal{X} \to \mathbb{R}$ with associated kernel function $k(\cdot,\cdot)$. Note that $f$ is both a function on $\mathcal{X}$ and a vector in $\mathcal{H}$. With an abuse of notation, we will not differentiate them unless necessary. Denote the RKHS norm of $\mathcal{H}$ by $\|f\|_{\mathcal{H}}$. Then MI-SVM can be rewritten as follows:

$$\min_{f\in\mathcal{H},\boldsymbol{\xi}} \frac{1}{2}\|f\|^2 + C\Big[\sum_{i\in\mathcal{I}_+}\xi_i + \sum_{j\in\mathcal{I}_i,i\in\mathcal{I}_-}\xi_{ij}\Big]$$
$$s.t.\ \max_{j\in\mathcal{I}_i}(f(\mathbf{x}_{ij})) \geq 1-\xi_i,\ \xi_i\geq 0,\ i\in\mathcal{I}_+ \tag{6}$$
$$-(f(\mathbf{x}_{ij})) \geq 1-\xi_{ij},\ \xi_{ij}\geq 0,\ j\in\mathcal{I}_i,i\in\mathcal{I}_-$$

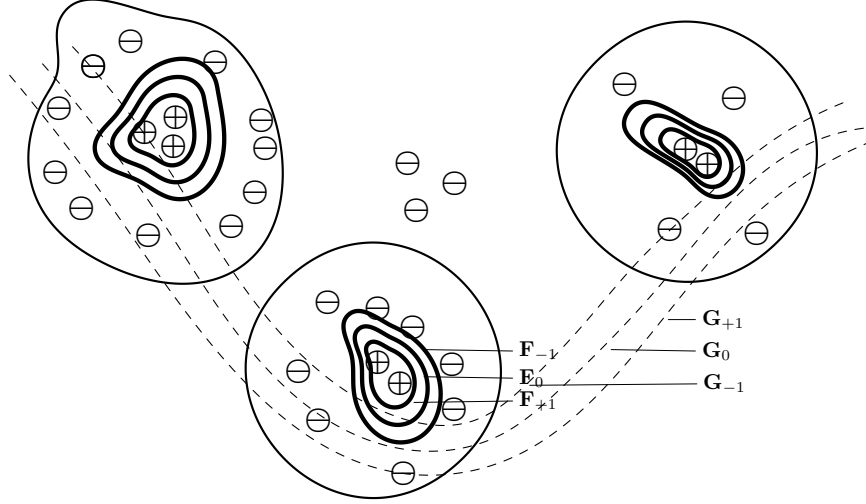

Figure 2: Illustration of the Effect of the Projection Constraint: Please note that the projection constraint is effective for datasets with any geometric distribution once an appropriate kernel is selected. Enveloped points are instances in a positive bag. Points not enveloped are negative bags of just one instance. Separating plane $\mathbf{F}_i$ corresponds to $f(\mathbf{x}) = i$, and $\mathbf{G}_i$ corresponds to $g(\mathbf{x}) = i$. The learner $f$ and $g$ are obtained with and without the projection constraint, respectively. Instances are labeled according to $f$. $\oplus$ and $\ominus$ denote positive instances and negative instance respectively.

According to the representer theorem [16], each minimizer $f \in \mathcal{H}$ of (6) has the following form:

$$f = \sum_{i \in \mathcal{I}_+ \cup \mathcal{I}_-} \sum_{j \in I_i} \alpha_{ij} \phi(\mathbf{x}_{ij}) \tag{7}$$

where all $\alpha_i \in \mathbb{R}$, and $\phi(\cdot)$ induced by $k(\cdot, \cdot)$ is the feature mapping from $\mathcal{X}$ to $\mathcal{H}$.

Next, we will propose our key contribution, i.e., the projection constraint. Given a positive bag $B_i$, denote its instances by $\{\mathbf{x}_{ij}\}_{j=1}^{n_i}$, and denote the normal vector of the separating plane in the RKHS by $f$. According to the theory of PCA [15, 17], maximizing the variance of mapped instances $\{\phi(\mathbf{x}_{ij})\}_{j=1}^{n_i}$ along $f$ equals to minimizing the sum of the Euclidean distances from the centralized data points to their projections on the normalized vector $\frac{f}{\|f\|_2}$, as follows:

$$J_i(f) = \sum_{j=1}^{n_i} \| c_j \frac{f}{\|f\|_2} - (\phi(\mathbf{x}_{ij}) - \phi(\mathbf{m_i})) \|_2^2 \tag{8}$$

where $\phi(\mathbf{m_i}) = \frac{1}{n_i} \sum_{j=1}^{n_i} \phi(\mathbf{x}_{ij})$, the mean of $\{\phi(\mathbf{x}_{ij})\}_{j=1}^{n_i}$. $|c_j|$ is the distance from $\phi(\mathbf{m_i})$ to the projection point of $\phi(\mathbf{x}_{ij})$. After simple algebra, we get:

$$c_j = \frac{f^T}{\|f\|_2} (\phi(\mathbf{x}_{ij}) - \phi(\mathbf{m_i})) \tag{9}$$

Substituting (9) and (7) into (8), we arrive at:

$$J_i(\boldsymbol{\alpha}) = o_i - \frac{\boldsymbol{\alpha}^T \overline{\mathbf{L}_i^2} \boldsymbol{\alpha}}{\boldsymbol{\alpha}^T \mathbf{K} \boldsymbol{\alpha}} \tag{10}$$

where $\mathbf{K}$ is a $n \times n$ kernel matrix defined on all the instances of both positive bags and negative bags, $o_i = trace(\mathbf{K}_{I(B_i)}) - \frac{1}{n_i} \mathbf{1}^T \mathbf{K}_{I(B_i)} \mathbf{1}$ where $\mathbf{K}_{I(B_i)}$ is a $n_i \times n_i$ matrix formed by extracting the $I(B_i)$ columns (Please refer to section 2.1) and the $I(B_i)$ rows of the overall kernel matrix $\mathbf{K}$, and $\overline{\mathbf{L}_i^2}$ is the "centralized" $\mathbf{L}_i^2$ as follows:

$$\overline{\mathbf{L}_i^2} = \mathbf{L}_i^T \mathbf{L}_i - \mathbf{1}_n \mathbf{L}_i^T \mathbf{L}_i - \mathbf{L}_i^T \mathbf{L}_i \mathbf{1}_n + \mathbf{1}_n \mathbf{L}_i^T \mathbf{L}_i \mathbf{1}_n \tag{11}$$

where $\mathbf{1}_n$ is a matrix with all elements equal to $\frac{1}{n}$, and $\mathbf{L}_i$ is a $n \times n$ matrix formed by keeping the $I(B_i)$ rows of $\mathbf{K}$ and setting all the elements in other rows to 0:

$$\mathbf{L}_i(p,q) = \begin{cases} \mathbf{K}(p,q) & \text{if } p \in I(B_i), \forall q \in \{1, \cdots, n\} \\ 0 & \text{otherwise} \end{cases}$$

Generally, the optimal normal vector $f$ varies for different positive bags. Hence it is meaningless to solve (10) for its optimum. Instead, we average (10) by the bag size $n_i$, and use a common threshold $\lambda$ to bound the averaged projection distance for different bags from above. We name the obtained inequality "the projection constraint", as follows:

$$\frac{1}{n_i}\left(o_i - \frac{\boldsymbol{\alpha}^T \overline{\mathbf{L}_i^2} \boldsymbol{\alpha}}{\boldsymbol{\alpha}^T \mathbf{K} \boldsymbol{\alpha}}\right) \leq \lambda \tag{12}$$

This is equivalent to bounding variance of instances in positive bags along $f$ from below [15].

Substituting (7) into (6), and adding the projection constraint (12) for each positive bag to the resulted problem, we arrive at the following optimization problem:

$$\min_{\boldsymbol{\alpha}, b, \boldsymbol{\xi}} \frac{1}{2} \boldsymbol{\alpha}^T \mathbf{K} \boldsymbol{\alpha} + C \Big[ \sum_{i \in \mathcal{I}_+} \xi_i + \sum_{j \in \mathcal{I}_i, i \in \mathcal{I}_-} \xi_{ij} \Big] \tag{13}$$

$$s.t. \ 1 - \xi_i - \max_{j \in \mathcal{I}_i}(\mathbf{k}_{I(\mathbf{x}_{ij})}^T \boldsymbol{\alpha} + b) \leq 0, \ \xi_i \geq 0, \ i \in \mathcal{I}_+$$

$$\mathbf{k}_{I(\mathbf{x}_{ij})}^T \boldsymbol{\alpha} + b \leq -1 + \xi_{ij}, \ \xi_{ij} \geq 0, \ j \in \mathcal{I}_i, i \in \mathcal{I}_-$$

$$\boldsymbol{\alpha}^T(o_i \cdot \mathbf{K} - \overline{\mathbf{L}_i^2})\boldsymbol{\alpha} - \lambda n_i \cdot \boldsymbol{\alpha}^T \mathbf{K} \boldsymbol{\alpha} \leq 0, \ i \in \mathcal{I}_+$$

### 3.3 Optimization via the CCCP

In the problem (13), the objective function and the second set of constraints are convex. The first set of constraints are all in the form of difference of two convex functions since the $\max$ function is convex. According to the definition of $J_i(f)$ in (8), $J(\boldsymbol{\alpha})$ in (10) is not less than 0 for any $\boldsymbol{\alpha}$. Thus for any $i \in \mathcal{I}_+$, $o_i \cdot \mathbf{K} - \overline{\mathbf{L}_i^2}$ is semi-definite positive. Consequently, the third set of constraints are all in the form of difference of two convex functions. Therefore, we can apply the Constrained Concave-Convex Procedure (CCCP) introduced in section 2.2 to solve the problem (13).

Since the function $\max$ in the first set of constraints is nonsmooth, we have to change gradients to subgradients to use the CCCP. The subgradient is usually not unique, and we adopt the definition used in [6] for the subgradient of $\max_{j \in \mathcal{I}_i} \mathbf{k}_{I(\mathbf{x}_{ij})}^T \boldsymbol{\alpha}$:

$$\partial(\max_{j \in \mathcal{I}_i} \mathbf{k}_{I(\mathbf{x}_{ij})}^T \boldsymbol{\alpha}) = \sum_{j \in \mathcal{I}_i} \beta_{ij} \mathbf{k}_{I(\mathbf{x}_{ij})}^T \tag{14}$$

where

$$\beta_{ij} = \begin{cases} 0 & \text{if } \mathbf{k}_{I(\mathbf{x}_{ij})}^T \boldsymbol{\alpha} \neq \max_{j \in \mathcal{I}_i} \mathbf{k}_{I(\mathbf{x}_{ij})}^T \boldsymbol{\alpha} \\ \frac{1}{n_a} & \text{otherwise} \end{cases} \tag{15}$$

where $n_a$ is the number of $\mathbf{x}_{ij}$ that maximize $\mathbf{k}_{I(\mathbf{x}_{ij})}^T \boldsymbol{\alpha}$. At the $t^{th}$ iteration, denote the current estimate for $\boldsymbol{\alpha}$ and $\beta_{ij}$ by $\boldsymbol{\alpha}^{(t)}$ and $\beta_{ij}^{(t)}$ respectively. Then the first order Taylor expansion of $\max_{j \in \mathcal{I}_i} \mathbf{k}_{I(\mathbf{x}_{ij})}^T \boldsymbol{\alpha}$ is as follows:

$$\max_{j \in \mathcal{I}_i} \mathbf{k}_{I(\mathbf{x}_{ij})}^T \boldsymbol{\alpha}^{(t)} + \sum_{j \in \mathcal{I}_i} \beta_{ij}^{(t)} \mathbf{k}_{I(\mathbf{x}_{ij})}^T (\boldsymbol{\alpha} - \boldsymbol{\alpha}^{(t)}) \tag{16}$$

According to (15), we have

$$\sum_{j \in \mathcal{I}_i} \beta_{ij}^{(t)} \mathbf{k}_{I(\mathbf{x}_{ij})}^T \boldsymbol{\alpha}^{(t)} = \max_{j \in \mathcal{I}_i}(\mathbf{k}_{I(\mathbf{x}_{ij})}^T \boldsymbol{\alpha}^{(t)}) \tag{17}$$

Using (17), (16) reduces to

$$\sum_{j \in \mathcal{I}_i} \beta_{ij}^{(t)} \mathbf{k}_{I(\mathbf{x}_{ij})}^T \boldsymbol{\alpha} \tag{18}$$

Replacing $\max_{j \in \mathcal{I}_i} \mathbf{k}_{I(\mathbf{x}_{ij})}^T \boldsymbol{\alpha}$ in the first set of constraints by (18) and $\boldsymbol{\alpha}^T \boldsymbol{K} \boldsymbol{\alpha}$ in the third set of constraints by their first order Taylor expansions, finally we get:

$$\min_{\boldsymbol{\alpha}, b, \boldsymbol{\xi}} \frac{1}{2} \boldsymbol{\alpha}^T \mathbf{K} \boldsymbol{\alpha} + C \Big[ \sum_{i \in \mathcal{I}_+} \xi_i + \sum_{i \in \mathcal{I}_i, I \in \mathcal{I}_-} \xi_i \Big] \tag{19}$$

$$s.t. \ 1 - \xi_i - \Big( \sum_{j \in \mathcal{I}_i} \beta_{ij}^{(t)} \mathbf{k}_{I(\mathbf{x}_{ij})}^T \boldsymbol{\alpha} + b \Big) \le 0, \ \xi_i \ge 0, \ i \in \mathcal{I}_+$$

$$\mathbf{k}_{I(\mathbf{x}_{ij})}^T \boldsymbol{\alpha} + b \le -1 + \xi_{ij}, \ \xi_{ij} \ge 0, \ j \in \mathcal{I}_i, i \in \mathcal{I}_-$$

$$\boldsymbol{\alpha}^T S_i \boldsymbol{\alpha} - 2\lambda n_i \cdot \boldsymbol{\alpha}^{(t)^T} K(\boldsymbol{\alpha} - \boldsymbol{\alpha}^{(t)}) \le 0, \ i \in \mathcal{I}_+$$

where $S_i = o_i \cdot \mathbf{K} - \overline{\mathbf{L}_i^2}$. The problem (19) is a quadratically constrained quadratic program (QCQP) with a convex objective function and convex constraints, and thus can be readily solved via interior point methods [18]. Following the CCCP, we can do the iteration until (19) converges.

## 4  Experiments

### 4.1  Classification: Benchmark

Benchmark data sets comes from two areas. Musk 1 and Musk 2 data sets [1] are two biochemical tasks which directly promoted the research of multi-instance learning. The aim is to predict activity of drugs from structural information. Each drug molecule is a bag of potential conformations (instances). The Musk 1 data set consists of 47 positive bags, 45 negative bags, and totally 476 instances. The Musk 2 data set consists of 39 positive bags, 63 negative bags, and totally 6598 instances. Each instance is represented by a 166 dimensional vector. Elephant, tiger and fox are three data sets from image categorization. The aim is to differentiate images with elephant, tiger, and fox [4] from those without, respectively. A bag here is a group of ROIs (Region Of Interests) drawn from a certain image. Each data set contains 100 positive bags and 100 negative bags, and each ROI as an instance is a 230 dimensional vector. Related methods for comparison includes Diverse

Table 1: Test Accuracy(%) On Benchmark: Rows and columns correspond to methods and datasets respectively.

| Algorithm | Musk 1 | Musk 2 | Elep | Fox | Tiger |
|---|---|---|---|---|---|
| PC-SVM | 90.6 ±2.7 | **91.3** ±**3.2** | **89.8** ±**1.2** | **65.7** ±**1.4** | 83.8 ±1.3 |
| MIGraph | 90.0 ±3.8 | 90.0 ±2.7 | 85.1 ±2.8 | 61.2 ±1.7 | 81.9 ±1.5 |
| miGraph | 88.9 ±3.3 | 90.3 ±2.6 | 86.8 ±0.7 | 61.6 ±2.8 | **86.0** ±**1.0** |
| MI-Kernel | 88.0 ±3.1 | 89.3 ±1.5 | 84.3 ±1.6 | 60.3 ±1.9 | 84.2 ±1.0 |
| MI-SVM | 77.9 | 84.3 | 81.4 | 59.4 | 84.0 |
| stMIL | 79.5 | 68.4 | 81.6 | 60.7 | 74.7 |
| sbMIL | **91.8** | 87.7 | 88.6 | 69.8 | 83.0 |
| DD | 88.0 | 84.0 | N/A | N/A | N/A |
| EM-DD | 84.8 | 84.9 | 78.3 | 56.1 | 72.1 |

Density (DD,[2]), EM-DD [19], MI-SVM [4], MI-Kernel [5], stMIL [8], sbMIL [8], MIGraph and miGraph [10]. When applied for multi-instance classification, our approach involves three parameters, namely, the bias/variance trade-off factor $C$, the kernel parameter (e.g.: $\gamma$ in RBF kernel), and the bound parameter $\lambda$ in the projection constraint. In the experiment, $C$, $\gamma$, and $\lambda$ are selected from

{0.01,0.1,1,10,50,100}, {0.2,0.4,0.6,0.8,1.0} and {0.01,0.1,1,10,100} respectively. We employ the MOSEK toolbox [1] to solve the resulted QCQP problem (19). The other experiment uses the same parameter setting.

The ten-times 10-fold cross validation results (except Diverse Density) are shown in Table 1. The results for other methods are replicated from their original papers. The results not available are marked by N/A. The bolded figure indicates that result is better than all other methods. Table 1 shows that the performance of our approach (PC-SVM) is competitive. Recall that the difference between our approach and MI-SVM is just the projection constraint. Therefore, as discussed in section 3.2, the results in Table 1 demonstrates that the strength of nonwitnesses is well utilized via the projection constraint.

### 4.2 Classification: COREL Image Data Sets

Table 2: Test Accuracy(%) On COREL: Rows and columns correspond to methods and datasets respectively.

| Algorithm | 1000-Image | 2000-Image |
|---|---|---|
| PC-SVM | **85.6 : [84.3, 86.9]** | **75.8 : [74.4, 77.2]** |
| reg-SVM | 84.4 : [83.0, 85.8] | N/A |
| MIGraph | 83.9 : [81.2, 85.7] | 72.1 : [71.0, 73.2] |
| miGraph | 82.4 : [80.2, 82.6] | 70.5 : [68.7, 72.3] |
| MI-Kernel | 81.8 : [80.1, 83.6] | 72.0 : [71.2, 72.8] |
| MI-SVM | 74.7 : [74.1, 75.3] | 54.6 : [53.1, 56.1] |
| DD-SVM | 81.5 : [78.5, 84.5] | 67.5 : [66.1, 68.9] |

COREL is a collection of natural scene images which have been categorized according to the presence of certain objects. Each image is regarded as a bag, and the nine dimensional ROIs (Region Of Interests) in it are regarded as its constituent instances. In experiments, we use the 1000-Image data set and the 2000-Image data set which contain ten and twenty categorizes, respectively. Following the methodology in [10], on both of the two data sets the related methods are compared by their five times 2-fold cross validation results. The algorithm for comparison include Diverse Density (DD), MI-SVM, MIGraph, miGraph , MI-Kernel and reg-SVM. In the last four algorithms one-against-all strategy is employed to tackle this multi-class task. In our approach this strategy is also used. Table 2 shows the overall accuracy as well as the $95\%$ interval. As in benchmark data sets, our approach is competitive with the latest methods. The results again suggest that fully utilizing the nonwitnesses is important for multi-instance classification.

## 5 Conclusion

We design a projection constraint to fully exploit nonwitnesses to avoid false positive. Since our approach is basically MI-SVM with projection constraints, the improved results on real world data sets validate the strength of nonwitnesses. We will introduce the universal projection constraint into other existing approaches for multi-instance learning, and related learning tasks, such as multi-instance regression, multi-label multi-instance learning, generalized multi-instance learning, etc.

**Acknowledgments**

We gratefully acknowledge reviewers for their insightful remarks and editors for their assiduous work. We also deeply appreciate Kuijun Ma's careful proof-reading. Finally, we are extremely thankful to Runing Liu for the fascinating illustrations. This work was partially supported by National Basic Research Program of China under Grant No.2004CB318103 and National Natural Science Foundation of China under award No.60835002 and 60975040.

## Footnotes

[1] http://www.mosek.com/

# References

[1] T. G. Dietterich, R. H. Lathrop, and T. Lozano-Pérez. Solving the multiple-instance problem with axis-parallel rectangles. *Artificial Intelligence*, 89(1-2):31–71, 1997.

[2] O. Maron and T. Lozano-Pérez. A framework for multiple-instance learning. *Advances in neural information processing systems*, pages 570–576, 1998.

[3] J. Wang and J.D. Zucker. Solving the multiple-instance problem: A lazy learning approach. In *Proceedings of the Seventeenth International Conference on Machine Learning*, pages 1119–1126. Citeseer, 2000.

[4] S. Andrews, I. Tsochantaridis, and T. Hofmann. Support vector machines for multiple-instance learning. *Advances in neural information processing systems*, pages 577–584, 2003.

[5] T. Gärtner, P.A. Flach, A. Kowalczyk, and A.J. Smola. Multi-instance kernels. In *Proceedings of the Nineteenth International Conference on Machine Learning*, pages 179–186. Citeseer, 2002.

[6] P.M. Cheung and J.T. Kwok. A regularization framework for multiple-instance learning. In *Proceedings of the 23rd international conference on Machine learning*, page 200. ACM, 2006.

[7] Z.H. Zhou and J.M. Xu. On the relation between multi-instance learning and semi-supervised learning. In *Proceedings of the 24th international conference on Machine learning*, page 1174. ACM, 2007.

[8] R.C. Bunescu and R.J. Mooney. Multiple instance learning for sparse positive bags. In *Proceedings of the 24th international conference on Machine learning*, page 112. ACM, 2007.

[9] H.Y. Wang, Q. Yang, and H. Zha. Adaptive p-posterior mixture-model kernels for multiple instance learning. In *Proceedings of the 25th international conference on Machine learning*, pages 1136–1143. ACM, 2008.

[10] Z. H. Zhou, Y. Y. Sun, and Yu. F. Li. Multi-instance learning by treating instances as non-I.I.D. samples. In Léon Bottou and Michael Littman, editors, *Proceedings of the 26th International Conference on Machine Learning*, pages 1249–1256, Montreal, June 2009. test, Omnipress.

[11] Y. Chen and J.Z. Wang. Image categorization by learning and reasoning with regions. *The Journal of Machine Learning Research*, 5:913–939, 2004.

[12] B. Settles, M. Craven, and S. Ray. Multiple-instance active learning. *Advances in Neural Information Processing Systems (NIPS)*, 20:1289–1296, 2008.

[13] G. Fung, M. Dundar, B. Krishnapuram, and R.B. Rao. Multiple instance learning for computer aided diagnosis. In *NIPS2007*, page 425. The MIT Press, 2007.

[14] A.J. Smola, SVN Vishwanathan, and T. Hofmann. Kernel methods for missing variables. In *Proceedings of the Tenth International Workshop on Artificial Intelligence and Statistics*. Citeseer, 2005.

[15] R.O. Duda, P.E. Hart, and D.G. Stork. *Pattern classification*. John Wiley & Sons, 2001.

[16] B. Schölkopf and A.J. Smola. *Learning with kernels*. Citeseer, 2002.

[17] Q. Tao, D.J. Chu, and J. Wang. Recursive support vector machines for dimensionality reduction. *IEEE Transactions on Neural Networks*, 19(1):189–193, 2008.

[18] S.P. Boyd and L. Vandenberghe. *Convex optimization*. Cambridge Univ Pr, 2004.

[19] Q. Zhang and S.A. Goldman. Em-dd: An improved multiple-instance learning technique. *Advances in neural information processing systems*, 2:1073–1080, 2002.

